# On the Algorithmics and Applications of a Mixed-norm based Kernel Learning Formulation

**J. Saketha Nath**
Dept. of Computer Science & Engg.,
Indian Institute of Technology, Bombay.
saketh@cse.iitb.ac.in

**G. Dinesh**
Dept. of Computer Science & Automation,
Indian Institute of Science, Bangalore.
dinesh@csa.iisc.ernet.in

**S. Raman**
Dept. of Computer Science & Automation,
Indian Institute of Science, Bangalore.
sraman@csa.iisc.ernet.in

**Chiranjib Bhattacharyya**
Dept. of Computer Science & Automation,
Indian Institute of Science, Bangalore.
chiru@csa.iisc.ernet.in

**Aharon Ben-Tal**
Faculty of Industrial Engg. & Management,
Technion, Haifa.
abental@ie.technion.ac.il

**K. R. Ramakrishnan**
Dept. of Electrical Engg.,
Indian Institute of Science, Bangalore.
krr@ee.iisc.ernet.in

## Abstract

Motivated from real world problems, like object categorization, we study a particular mixed-norm regularization for Multiple Kernel Learning (MKL). It is assumed that the given set of kernels are grouped into distinct components where each component is crucial for the learning task at hand. The formulation hence employs $l_\infty$ regularization for promoting combinations at the component level and $l_1$ regularization for promoting sparsity among kernels in each component. While previous attempts have formulated this as a non-convex problem, the formulation given here is an instance of non-smooth convex optimization problem which admits an efficient Mirror-Descent (MD) based procedure. The MD procedure optimizes over product of simplexes, which is not a well-studied case in literature. Results on real-world datasets show that the new MKL formulation is well-suited for object categorization tasks and that the MD based algorithm outperforms state-of-the-art MKL solvers like simpleMKL in terms of computational effort.

## 1 Introduction

In this paper the problem of Multiple Kernel Learning (MKL) is studied where the given kernels are assumed to be grouped into distinct components and each component is crucial for the learning task in hand. The focus of this paper is to study the formalism, algorithmics of a specific mixed-norm regularization based MKL formulation suited for such tasks.

Majority of existing MKL literature have considered employing a block $l_1$ norm regularization leading to selection of few of the given kernels [8, 1, 16, 14, 20] . Such formulations tend to select the "best" among the given kernels and consequently the decision functions tend to depend only on the selected kernel. Recently [17] extended the framework of MKL to the case where kernels are partitioned into groups and introduces a generic mixed-norm regularization based MKL formulation in order to handle groups of kernels. Again the idea is to promote sparsity leading to low number of kernels. This paper differs from [17] by assuming that every component (group of kernels) is highly

crucial for success of the learning task. It is well known in optimization literature that $l_\infty$ regularizations often promote combinations with equal preferences and $l_1$ regularizations lead to selections. The proposed MKL formulation hence employs $l_\infty$ regularization and promotes combinations of kernels at the component level. Moreover it employs $l_1$ regularization for promoting sparsity among kernels in each component.

The formulation studied here is motivated by real-world learning applications like object categorization where multiple feature representations need to be employed simultaneously for achieving good generalization. Combining feature descriptors using the framework of Multiple Kernel Learning (MKL) [8] for object categorization has been a topic of interest for many recent studies [19, 13]. For e.g., in the case of flower classification feature descriptors for shape, color and texture need to be employed in order to achieve good visual discrimination as well as significant within-class variation [12]. A key finding of [12] is the following: in object categorization tasks, employing few of the feature descriptors or employing a canonical combination of them often leads to sub-optimal solutions. Hence, in the framework of MKL, employing a $l_1$ regularization, which is equivalent to selecting one of the given kernels, as well as employing a $l_2$ regularization, which is equivalent to working with a canonical combination of the given kernels, may lead to sub-optimality. This important finding clearly motivates the use of $l_\infty$ norm regularization for combining kernels generated from different feature descriptors and $l_1$ norm regularization for selecting kernels generated from the same feature descriptor. Hence, by grouping kernels generated from the same feature descriptor together and employing the new MKL formulation, classifiers which are potentially well-suited for object categorization tasks can be built.

Apart from the novel MKL formulation the main contribution of the paper is a highly efficient algorithm for solving it. Since the formulation is an instance of a Second Order Cone Program (SOCP), it can be solved using generic interior point algorithms. However it is impractical to work with such solvers even for moderately large number of data points and kernels. Also the generic wrapper approach proposed in [17] cannot be employed as it solves a non-convex variant of the proposed (convex) formulation. The proposed algorithm employs mirror-descent [3, 2, 9] leading to extremely scalable solutions.

The feasibility set for the minimization problem tackled by Mirror-Descent (MD) turns out to be direct product of simplexes, which is not a standard set-up discussed in optimization literature. We employ a weighted version of the entropy function as the prox-function in the auxiliary problem solved by MD at each iteration and justify its suitability for the case of direct product of simplexes. The mirror-descent based algorithm presented here is also of independent interest to the MKL community as it can solve the traditional MKL problem; namely the case when the number of groups is unity. Empirically we show that the mirror-descent based algorithm proposed here scales better than the state-of-the-art steepest descent based algorithms [14].

The remainder of this paper is organized as follows: in section 2, details of the new MKL formulation and its dual are presented. The mirror-descent based algorithm which efficiently solves the dual is presented in section 3. This is followed by a summary of the numerical experiments carried for verifying the major claims of the paper. In particular, the empirical findings are a) the new MKL formulation is well-suited for object categorization tasks b) the MD based algorithm scales better than state-of-the-art gradient descent methods (e.g. `simpleMKL`) in solving the special case where number of components (groups) of kernels is unity.

## 2 Mixed-norm based MKL Formulation

This section presents the novel mixed-norm regularization based MKL formulation and its dual. In the following text we concentrate on the case of binary classification. However many of the ideas presented here apply to other learning problems too. Let the training dataset be denoted by $\mathcal{D} = \{(\mathbf{x}_i, y_i), \ i = 1, \ldots, m \mid \mathbf{x}_i \in \mathcal{X}, \ y_i \in \{-1, 1\}\}$. Here, $\mathbf{x}_i$ represents the $i^{th}$ training data point with label $y_i$. Let $\mathbf{Y}$ denote the diagonal matrix with entries as $y_i$. Suppose the given kernels are divided into $n$ groups (components) and the $j^{th}$ component has $n_j$ number of kernels. Let the feature-space mapping generated from the $k^{th}$ kernel of the $j^{th}$ component be $\phi_{jk}(\cdot)$ and the corresponding gram-matrix of training data points be $\mathbf{K}_{jk}$[1]. We are in search of a hyperplane clas-

sifier of the form $\sum_{j=1}^n \sum_{k=1}^{n_j} \mathbf{w}_{jk}^\top \phi_{jk}(\mathbf{x}_i) - b = 0$. As discussed above, we wish to perform a block $l_\infty$ regularization over the model parameters $\mathbf{w}_{jk}$ associated with distinct components and $l_1$ regularization for those associated with the same component. Intuitively, such a regularization promotes combinations of kernels belonging to different components and selections among kernels of the same component. Following the framework of MKL and the mixed norm regularization detailed here, the following formulation is immediate:

$$\min_{\mathbf{w}_{jk}, b, \xi_i} \quad \frac{1}{2}\left[\max_j\left(\sum_{k=1}^{n_j}\|\mathbf{w}_{jk}\|_2\right)^2\right] + C\sum_i \xi_i$$

$$\text{s.t.} \quad y_i\left(\sum_{j=1}^n\sum_{k=1}^{n_j}\mathbf{w}_{jk}^\top\phi_{jk}(\mathbf{x}_i) - b\right) \geq 1 - \xi_i,\ \xi_i \geq 0 \ \forall\ i \tag{1}$$

Here, $\xi_i$ variables measure the slack in correctly classifying the $i^{th}$ training data point and $C$ is the regularization parameter controlling weightage given to the mixed-norm regularization term and the total slack. MKL formulation in (1) is convex and moreover an instance of SOCP. This formulation can also be realized as a limiting case of the generic CAP formulation presented in [17] (with $\gamma = 1, \gamma_0 \to \infty$). However since the motivation of that work was to perform feature selection, this limiting case was neither theoretically studied nor empirically evaluated. Moreover, the generic wrapper approach of [17] is inappropriate for solving this limiting case as that approach would solve a non-convex variant of this (convex) formulation. In the following text, a dual of (1) is derived.

Let a simplex of dimensionality $d$ be represented by $\Delta_d$. Following the strategy of [14], one can introduce variables $\lambda_j \equiv \left[\lambda_{j1}\ldots\lambda_{jn_j}\right]^\top \in \Delta_{n_j}$ and re-write (1) as follows:

$$\min_{\mathbf{w}_{jk}, b, \xi_i} \quad \frac{1}{2}\left[\max_j\left(\min_{\lambda_j \in \Delta_{n_j}}\sum_{k=1}^{n_j}\frac{\|\mathbf{w}_{jk}\|_2^2}{\lambda_{jk}}\right)\right] + C\sum_i \xi_i$$

$$\text{s.t.} \quad y_i\left(\sum_{j=1}^n\sum_{k=1}^{n_j}\mathbf{w}_{jk}^\top\phi_{jk}(\mathbf{x}_i) - b\right) \geq 1 - \xi_i,\ \xi_i \geq 0 \ \forall\ i \tag{2}$$

This is because for any vector $[a_1\ldots a_n] \geq \mathbf{0}$, the following holds: $\min_{x_i \geq 0, \sum_i x_i = 1}\sum_i\frac{a_i^2}{x_i} = (\sum_i a_i)^2$. Notice that the $\max$ over j and $\min$ over $\lambda_j$ can be interchanged. To see that rewrite $\max_j$ as $\min_t t$ with constraints $\min_{\lambda_j \in \Delta_{n_j}}\sum_{k=1}^{n_j}\frac{\|\mathbf{w}_{jk}\|_2^2}{\lambda_{jk}} \leq t$, where $t$ is a new decision variable. This problem is feasible in both $\lambda_j$s and $t$ and hence we can drop the minimization over individual constraints to obtain an equivalent problem: $\min_{\lambda_j \in \Delta_{n_j} \forall j, t} t$ subject to $\sum_{k=1}^{n_j}\frac{\|\mathbf{w}_{jk}\|_2^2}{\lambda_{jk}} \leq t$. One can now eliminate $t$ by reintroducing the $\max_j$ and interchange the $\min_{\lambda_j \in \Delta_{n_j} \forall j}$ with other variables to obtain:

$$\min_{\lambda_j \in \Delta_{n_j} \forall j}\quad \min_{\mathbf{w}_{jk}, b, \xi_i} \quad \frac{1}{2}\max_j\sum_{k=1}^{n_j}\frac{\|\mathbf{w}_{jk}\|_2^2}{\lambda_{jk}} + C\sum_i \xi_i$$

$$\text{s.t.} \quad y_i\left(\sum_{j=1}^n\sum_{k=1}^{n_j}\mathbf{w}_{jk}^\top\phi_{jk}(\mathbf{x}_i) - b\right) \geq 1 - \xi_i,\ \xi_i \geq 0 \ \forall\ i \tag{3}$$

Now one can derive the standard dual of (3) wrt. to the variables $\mathbf{w}_{jk}, b, \xi_i$ alone, leading to:

$$\min_{\lambda_j \in \Delta_{n_j} \forall j}\quad \max_{\alpha \in S_m(C),\ \gamma \in \Delta_n} \quad \mathbf{1}^\top\alpha - \frac{1}{2}\alpha^\top\left[\sum_{j=1}^n\left(\frac{\sum_{k=1}^{n_j}\lambda_{jk}\mathbf{Q}_{jk}}{\gamma_j}\right)\right]\alpha \tag{4}$$

where $\alpha, \gamma$ are Lagrange multipliers, $S_m(C) \equiv \{\mathbf{x} \in \mathbb{R}^m \mid \mathbf{0} \leq \mathbf{x} \leq C\mathbf{1},\ \sum_{i=1}^m x_i y_i = 0\}$ and $\mathbf{Q}_{jk} \equiv \mathbf{Y}\mathbf{K}_{jk}\mathbf{Y}$. The following points regarding (4) must to be noted:

- (4) is equivalent to the well-known SVM [18] formulation with kernel $\mathbf{K}_{eff} \equiv \sum_{j=1}^n\left(\frac{\sum_{k=1}^{n_j}\lambda_{jk}^*\mathbf{K}_{jk}}{\gamma_j^*}\right)^2$. In other words, $\frac{1}{\gamma_j^*}$ is the weight given to the $j^{th}$ component and $\lambda_{jk}^*$ is weight given to the $k^{th}$ kernel of the $j^{th}$ component.
- It can be shown that none of $\gamma_j, j = 1, \ldots, n$ can be zero provided the given gram-matrices $\mathbf{K}_{jk}$ are positive definite[3].

- By construction, most of the weights $\lambda_{jk}$ are zero and at-least for one kernel in every component the weight is non-zero (see also [14]).

These facts readily justify the suitability of the particular mixed norm regularization for object categorization. Indeed, in-sync with findings of [12], kernels from different feature descriptors (components) are combined using non-trivial weights (i.e. $\frac{1}{\gamma_j^*}$). Moreover, only the "best" kernels from each feature descriptor (component) are utilized by the model. This sparsity feature leads to better interpretability as well as computational benefits during the prediction stage. In the following section an efficient iterative algorithm for solving the dual (4) is presented.

## 3 Efficient Algorithm for Solving the Dual

This section presents an efficient algorithm for solving the dual (4). Note that typically in object categorization or other such multi-modal learning tasks, the number of feature descriptors (i.e. number of groups of kernels, $n$) is low ($< 10$). However the kernels constructed from each feature descriptor can be very high in number i.e., $n_j \ \forall \ j$ can be quite high. Also, it is frequent to encounter datasets with huge number of training data points, $m$. Hence it is desirable to derive algorithms which scale well wrt. $m$ and $n_j$. We assume $n$ is small and almost $O(1)$. Consider the dual formulation (4). Using the minimax theorem [15], one can interchange the $\min$ over $\lambda_j$s and $\max$ over $\gamma$ to obtain:

$$\min_{\gamma \in \Delta_n} - \underbrace{\left[ \underbrace{\min_{\lambda_j \in \Delta_{n_j} \forall j} \left\{ \max_{\alpha \in S_m(C)} \mathbf{1}^\top \alpha - \frac{1}{2} \alpha^\top \left[ \sum_{j=1}^n \left( \frac{\sum_{k=1}^{n_j} \lambda_{jk} \mathbf{Q}_{jk}}{\gamma_j} \right) \right] \alpha \right\}}_{g_\gamma(\lambda_1,\dots,\lambda_n)} \right]}_{f(\gamma)} \tag{5}$$

We have restated the maximum over $\gamma$ as a minimization problem by introducing a minus sign. The proposed algorithm performs alternate minimization over the variables $\gamma$ and $(\lambda_1, \dots, \lambda_n, \alpha)$. In other words, in one step the variables $(\lambda_1, \dots, \lambda_n, \alpha)$ are assumed to be constant and (5) is optimized wrt. $\gamma$. This leads to the following optimization problem:

$$\min_{\gamma \in \Delta_n} \sum_{j=1}^n \frac{W_j}{\gamma_j}$$

where $W_j = \alpha^\top \sum_{k=1}^{n_j} \lambda_{jk} \mathbf{Q}_{jk} \alpha$. This problem has an analytical solution given by:

$$\gamma_j = \frac{\sqrt{W_j}}{\sum_j \sqrt{W_j}} \tag{6}$$

In the subsequent step $\gamma$ is assumed to be fixed and (5) is optimized wrt. $(\lambda_1, \dots, \lambda_n, \alpha)$. For this $f(\gamma)$ needs to be evaluated by solving the corresponding optimization problem (refer (5) for definition of $f$). Now, the per-step computational complexity of the iterative algorithm will depend on how efficiently one evaluates $f$ for a given $\gamma$. In the following we present a mirror-descent (MD) based algorithm which evaluates $f$ to sufficient accuracy in $O(\log [\max_j n_j]) O(\text{SVM}_m)$. Here $O(\text{SVM}_m)$ represents the computational complexity of solving an SVM with $m$ training data points. Neglecting the $\log$ term, the overall per-step computational effort for the alternate minimization can be assumed to be $O(\text{SVM}_m)$ and hence nearly-independent of the number of kernels. Alternatively, one can employ the strategy of [14] and compute $f$ using projected steepest-descent (SD) methods. The following points highlight the merits and de-merits of these two methods:

- In case of SD, the per-step auxiliary problem has no closed form solution and projections onto the feasibility set need to be done which are computationally intensive especially for problems with high dimensions. In case of MD, the auxiliary problem has an analytical solution (refer (8)).
- The step size needs to be computed using 1-d line search in case of SD; whereas the step-sizes for MD can be easily computed using analytical expressions (refer (9)).

- The computational complexity of evaluating $f$ using MD is nearly-independent of no. kernels. However no such statement can be made for SD (unless feasibility set is of Euclidean geometry, which is not so in our case).

The MD based algorithm for evaluating $f(\gamma)$ i.e. solving $\min_{\lambda_j \in \Delta_{n_j} \forall j} g_\gamma(\lambda_1, \ldots, \lambda_n)$ is detailed below. Let $\lambda$ represent the vector $[\lambda_1 \ldots \lambda_n]^\top$. Also let values at iteration 't' be indicated using the super-script '(t)'. Similar to any gradient-based method, at each step 't' MD works with a linear approximation of $g_\gamma$: $\hat{g}_\gamma^{(t)}(\lambda) = g_\gamma(\lambda^{(t)}) + (\lambda - \lambda^{(t)})^\top \nabla g_\gamma(\lambda^{(t)})$ and follows the below update rule:

$$\lambda^{(t+1)} = argmin_{\lambda \in \Delta_{n_1} \times \ldots \times \Delta_{n_n}} \left[ \hat{g}_\gamma^{(t)}(\lambda) + \frac{1}{s_t} \omega(\lambda^{(t)}, \lambda) \right] \tag{7}$$

where, $\omega(x, y) \equiv \omega(y) - \omega(x) - (y - x)^\top \nabla \omega(x)$ is the Bregman-divergence (prox-function) associated with $\omega(x)$, a continuously differentiable strongly convex distance-generating function. $s_t$ is a regularization parameter and also determines the step-size. (7) is usually known as the auxiliary problem and needs to be solved at each step. Intuitively (7) minimizes a weighted sum of the local linear approximation of the original objective and a regularization term that penalizes solutions far from the current iterate. It is easy to show that the update rule in (7) leads to the SD technique if $\omega(x) = \frac{1}{2}\|x\|_2^2$ and step-size is chosen using 1-d line search. The key idea in MD is to choose the distance-generating function based on the feasibility set, which in our case is direct product of simplexes, such that (7) is very easy to solve. Note that for SD, with feasibility set as direct product of simplexes, (7) is not easy to solve especially in higher dimensions.

We choose the distance-generating function as the following modified entropy function: $\omega(x) \equiv \sum_{j=1}^{n} \sum_{k=1}^{n_j} \left( x_{jk} n^{-1} + \delta n^{-1} n_j^{-1} \right) \log \left( x_{jk} n^{-1} + \delta n^{-1} n_j^{-1} \right)$ where $\delta$ is a small positive number (say, $10e - 16$). Now, let $\tilde{g}_\gamma^{(t)} \equiv s_t \nabla g_\gamma(\lambda^{(t)}) - \nabla \omega(\lambda^{(t)})$. Note that $g_\gamma$ is nothing but the optimal objective of SVM with kernel $\mathbf{K}_{eff}$. Since it is assumed that each given kernel is positive definite, the optimal of the SVM is unique and hence gradient of $g_\gamma$ wrt. $\lambda$ exists [5]. Gradient of $g_\gamma$ can be computed using $\frac{\partial g_\gamma}{\partial \lambda_{jk}^{(t)}} = -\frac{1}{2} \frac{\left( \alpha^{(t)} \right)^\top \mathbf{Q}_{jk} \alpha^{(t)}}{\gamma_j}$ where $\alpha^{(t)}$ is the optimal $\alpha$ obtained by solving an SVM with kernel as $\sum_{j=1}^{n} \left( \frac{\sum_{k=1}^{n_j} \lambda_{jk}^{(t)} \mathbf{K}_{jk}}{\gamma_j} \right)$. With this notation, it is easy to show that the optimal update (7) has the following analytical form[4]:

$$\lambda_{jk}^{(t+1)} = \frac{\exp \left\{ -\tilde{g}_{\gamma jk}^{(t)} n \right\}}{\sum_{k=1}^{n_j} \exp \left\{ -\tilde{g}_{\gamma jk}^{(t)} n \right\}} \tag{8}$$

The following text discusses the convergence issues with MD. Let the modulus of strong convexity of $\omega$ wrt. $\| \cdot \| \equiv \| \cdot \|_1$ be $\sigma$. Also, let the $\omega$-size of feasibility set be defined as follows: $\Theta \equiv \max_{u,v \in \Delta_{n_1} \times \ldots \times \Delta_{n_n}} \omega(u, v)$. It is easy to verify that $\sigma = O(1)n^{-2}$ and $\Theta = O\left(\log[\max_j n_j]\right)$ in our case. The convergence and its efficiency follow from this result [3, 2, 9]:

**Result 1** *With step-sizes:* $s_t = \frac{\sqrt{\Theta \sigma}}{\|\nabla g_\gamma\|_* \sqrt{t}}$ *one has the following bound on error after iteration* $T$: $\epsilon_T = \min_{t \leq T} g_\gamma(\lambda^{(t)}) - g_\gamma(\lambda^*) \leq O(1) \frac{\sqrt{\Theta} L_{\|\cdot\|}(g_\gamma)}{\sqrt{\sigma T}}$

where $\| \cdot \|_*$ is the dual norm of the norm wrt. which the modulus of strong convexity was computed (in our case $\| \cdot \|_* = \| \cdot \|_\infty$) and $L_{\|\cdot\|}(h)$ is Lipschitz constant of function $h$ wrt. norm $\| \cdot \|$ (in our case $\| \cdot \| = \| \cdot \|_1$ and it can be shown that the Lipschitz constant exists for $g_\gamma$). Substituting the particular values for our case, we obtain

$$s_t = \frac{\sqrt{\log[\max_j n_j]}}{n\|\nabla g_\gamma\|_\infty \sqrt{t}} \tag{9}$$

and $\epsilon_T \propto \frac{\sqrt{\log[\max_j n_j]}}{\sqrt{T}}$. In other words, for reaching a reasonable approximation of the optimal, the number iterations required are $O(\log[\max_j n_j])$, which is nearly-independent of the number

of kernels. Since the computations in each iteration are dominated by the SVM optimization, the overall complexity of MD is (nearly) $O(SVM_m)$. Note that the iterative algorithm can be improved by improving the algorithm for solving the SVM problem. The overall algorithm is summarized in algorithm 1[5]. The MKL formulation presented here exploits the special structure in the kernels and

---

**Algorithm 1**: Mirror-descent based alternate minimization algorithm

---

**Data**: Labels and gram-matrices of training eg., component-id of each kernel, regularization parameter (C)
**Result**: Optimal values of $\alpha, \gamma, \lambda$ in (4)
**begin**
    Set $\gamma, \lambda$ to some initial feasible values.
    **while** *stopping criteria for $\gamma$ is not met* **do**    `/* Alternate minimization loop */`
        **while** *stopping criteria for $\lambda$ is not met* **do**    `/* Mirror-descent loop */`
            Solve SVM with current kernel weights and update $\alpha$
            Compute $\tilde{g}_\gamma^{(t)}$ and update $\lambda$ using (8)
        Compute $W_j$ and update $\gamma$ using (6)
    Return values of $\alpha, \gamma, \lambda$
**end**

---

leads to non-trivial combinations of the kernels belonging to different components and selections among the kernels of the same component. Moreover the proposed iterative algorithm solves the formulation with a per-step complexity of (almost) $O(SVM_m)$, which is the same as that with traditional MKL formulations (which do not exploit this structure). As discussed earlier, this efficiency is an outcome of employing state-of-the-art mirror-descent techniques. The MD based algorithm presented here is of independent interest to the MKL community. This is because, in the special case where number of components is unity (i.e. $n = 1$), the proposed algorithm solves the traditional MKL formulation. And clearly, owing to the merits of MD over SD discussed earlier, the new algorithm can potentially be employed to boost the performance of state-of-the-art MKL algorithms. Our empirical results confirm that the proposed algorithm (with $n = 1$) outperforms `simpleMKL` in terms of computational efficiency.

## 4 Numerical Experiments

This section presents results of experiments which empirically verify the major claims of the paper: a) The proposed formulation is well-suited for object categorization b) In the case $n = 1$, the proposed algorithm outperforms `simpleMKL` wrt. computational effort. In the following, the experiments done on real-world object categorization datasets are summarized. The proposed MKL formulation is compared with state-of-the-art methodology for object categorization [19, 13] that employs a block $l_1$ regularization based MKL formulation with additional constraints for including prior information regarding weights of kernels. Since such constraints lead to independent improvements with all formulations, the experiments here compare the following three MKL formulations without the additional constraints: **MixNorm-MKL**, the $(l_\infty, l_1)$ mixed-norm based MKL formulation studied in this paper; **L1-MKL**, the block $l_1$ regularization based MKL formulation [14]; and **L2-MKL**, which is nothing but an SVM built using the canonical combination of all kernels i.e. $\mathbf{K}_{eff} \equiv \sum_{j=1}^{n} \sum_{k=1}^{n_j} \mathbf{K}_{jk}$. In case of **MixNorm-MKL**, the MD based algorithm (section 3) was used to solve the formulation. The SVM problem arising at each step of mirror-descent is solved using the `libsvm` software[6]. **L1-MKL** is solved using `simpleMKL`[7]. **L2-MKL** is solved using `libsvm` and serves as a baseline for comparison. In all cases, the hyper-parameters of the various formulations were tuned using suitable cross-validation procedures and the accuracies reported denote testset accuracies achieved by the respective classifiers using the tuned set of hyper-parameters.

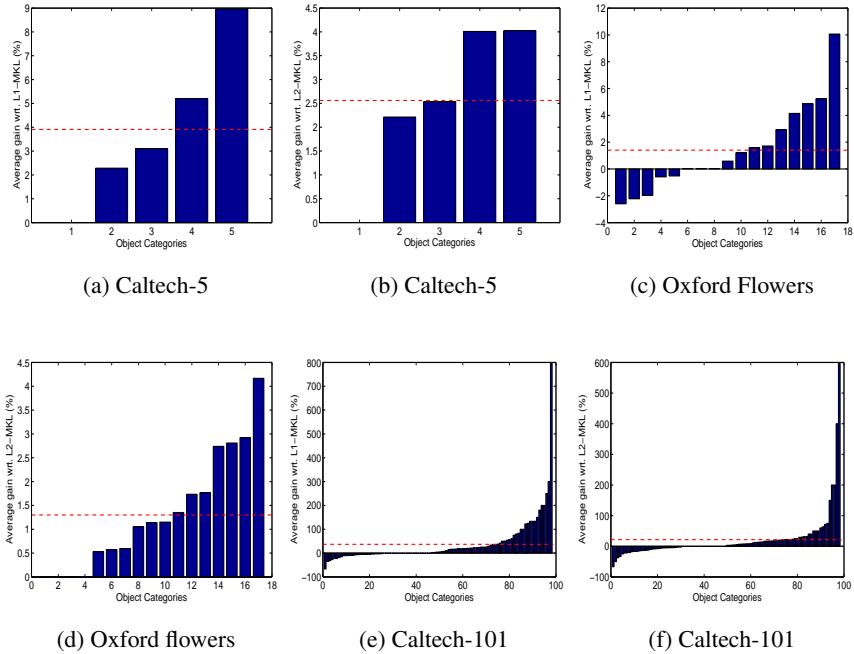

| (a) Caltech-5 | (b) Caltech-5 | (c) Oxford Flowers |
| (d) Oxford flowers | (e) Caltech-101 | (f) Caltech-101 |

Figure 1: Plot of average gain (%) in accuracy with **MixNorm-MKL** on the various real-world datasets.

The following real-world datasets were used in the experiments: Caltech-5 [6], Caltech-101 [7] and Oxford Flowers [10]. The Caltech datasets contain digital images of various objects like faces, watches, ants etc.; whereas the Oxford dataset contains images of 17 varieties of flowers. The Caltech-101 dataset has 101 categories of objects whereas Caltech-5 dataset is a subset of the Caltech-101 dataset including images of Airplanes, Car sides, Faces, Leopards and Motorbikes alone. Most categories of objects in the Caltech dataset have 50 images. The number of images per category varies from 40 to 800. In the Oxford flowers dataset there are 80 images in each flower category. In order to make the results presented here comparable to others in literature we have followed the usual practice of generating training and test sets using a fixed number of pictures from each object category and repeating the experiments with different random selections of pictures. For the Caltech-5, Caltech-101 and Oxford flowers datasets we have used 50, 15, 60 images per object category as training images and 50, 15, 20 images per object category as testing images respectively. Also, in case of Caltech-5 and Oxford flowers datasets, the accuracies reported are the testset accuracies averaged over 10 such randomly sampled training and test datasets. Since the Caltech-101 dataset has large number of classes and the experiments are computationally intensive (100 choose 2 classifiers need to be built in each case), the results are averaged over 3 sets of training and test datasets only. In case of the Caltech datasets, five feature descriptors[8] were employed: SIFT, OpponentSIFT, rgSIFT, C-SIFT, Transformed Color SIFT. Whereas in case of Oxford flowers dataset, following strategy of [11, 10], seven feature descriptors[9] were employed. Using each feature descriptor, nine kernels were generated by varying the width-parameter of the Gaussian kernel. The kernels can be grouped based on the feature descriptor they were generated from and the proposed formulation can be employed to construct classifiers well-suited for object categorization. For eg. in case of the Caltech datasets, $n = 5$ and $n_j = 9 \, \forall \, j$ and in case of Oxford flowers dataset, $n = 7$ and $n_j = 9 \, \forall \, j$. In all cases, the 1-vs-1 methodology was employed to handle the multi-class problems.

The results of the experiments are summarized in figure 1. Each plot shows the % gain in accuracy achieved by **MixNorm-MKL** over **L1-MKL** and **L2-MKL** for each object category. Note that for

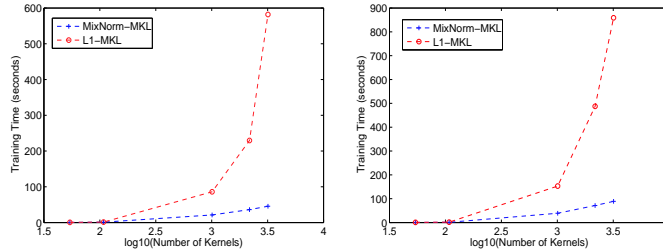

Figure 2: Scaling plots comparing scalability of mirror-descent based algorithm and `simpleMKL`.

most object categories, the gains are positive and moreover quite high. The best results are seen in case of the Caltech-101 dataset: the peak and avg. gains over **L1-MKL** are $800\%, 37.57\%$ respectively and over **L2-MKL** are $600\%, 21.75\%$ respectively. The gain in terms of numbers for the other two datasets are not as high merely because the baseline accuracies were themselves high. The baseline accuracies i.e., the average accuracy achieved by **L2-MKL** (over all categories) were $93.84\%, 34.81\%$ and $85.97\%$ for the Caltech-5, Caltech-101 and Oxford flowers datasets respectively. The figures clearly show that the proposed formulation outperforms state-of-the-art object categorization techniques and is hence highly-suited for such tasks. Another observation was that the average sparsity (% of kernels with zero weightages) with the methods **MixNorm-MKL, L1-MKL** and **L2-MKL** is $57\%, 96\%$ and $0\%$ respectively. Also, it was observed that **L1-MKL** almost always selected kernels from one or two components (feature descriptors) only whereas **MixNorm-MKL** (and ofcourse **L2-MKL**) selected kernels from all the components. These observations clearly show that the proposed formulation combines important kernels while eliminating redundant and noisy kernels using the information embedded in the group structure of the kernels.

In the following, the results of experiments which compare the scalability of `simpleMKL` and the proposed mirror-descent based algorithm wrt. the number of kernels are presented. Note that in the special case, $n = 1$, the proposed formulation is exactly same as the $l_1$ regularization based formulation. Hence the mirror-descent based iterative algorithm proposed here can also be employed for solving $l_1$ regularization based MKL. Figure 2 shows plots of the training times as a function of number of kernels with the algorithms on two binary classification problems encountered in the object categorization experiments. The plots clearly show that the proposed algorithm outperforms `simpleMKL` in terms of computational effort. Interestingly, it was found in our experiments that, in most cases, the major computational effort at every iteration of SimpleMKL was in computing the projection onto the feasible set! On the contrary Mirror descent allows an easily computable closed form solution for the per-step auxiliary problem. We think this is the crucial advantage of the proposed iterative algorithm over the gradient-decent based algorithms which were traditionally employed for solving the MKL formulations.

## 5   Conclusions

This paper makes two important contributions: a) a specific mixed-norm regularization based MKL formulation which is well-suited for object categorization and multi-modal tasks b) An efficient mirror-descent based algorithm for solving the new formulation. Empirical results on real-world datasets show that the new formulation achieves far better generalization than state-of-the-art object categorization techniques. In some cases, the average gain in testset accuracy compared to state-of-the-art was as high as $37\%$. The mirror-descent based algorithm presented in the paper not only solves the proposed formulation efficiently but also outperforms `simpleMKL` in solving the traditional $l_1$ regularization based MKL. The speed-up was as high as $12$ times in some cases. Application of proposed methodology to various other multi-modal tasks and study of improved variants of mirror-decent algorithm [4] for faster convergence are currently being explored by us.

**Acknowledgements** CB was supported by grants from Yahoo! and IBM.

## Footnotes

[1]The gram-matrices are unit-trace normalized.

[2] Superscript '*' represents the optimal value as per (4)

[3] Add a small ridge if positive semi-definite.

[4]Since the term involving $\delta$ is $\ll \lambda_{jk}$, it is neglected in this computation.

[5]Asymptotic convergence can be proved for the algorithm; details omitted due to lack of space.

[6]Available at `www.csie.ntu.edu.tw/~cjlin/libsvm`

[7]Available at `http://asi.insa-rouen.fr/enseignants/~arakotom/code/mklindex.html`

[8]Code at `http://staff.science.uva.nl/~ksande/research/colordescriptors/`

[9]Distance matrices available at `http://www.robots.ox.ac.uk/~vgg/data/flowers/17/index.html`

# References

[1] F. Bach, G. R. G. Lanckriet, and M. I. Jordan. Multiple Kernel Learning, Conic Duality, and the SMO Algorithm. In *International Conference on Machine Learning*, 2004.

[2] Amir Beck and Marc Teboulle. Mirror descent and nonlinear projected subgradient methods for convex optimization. *Operations Research Letters*, 31:167–175, 2003.

[3] Aharon Ben-Tal, Tamar Margalit, and Arkadi Nemirovski. The Ordered Subsets Mirror Descent Optimization Method with Applications to Tomography. *SIAM Journal of Optimization*, 12(1):79–108, 2001.

[4] Aharon Ben-Tal and Arkadi Nemirovski. Non-euclidean Restricted Memory Level Method for Large-scale Convex Optimization. *Mathematical Programming*, 102(3):407–456, 2005.

[5] O. Chapelle, V. Vapnik, O. Bousquet, and S. Mukerjhee. Choosing multiple parameters for SVM. *Machine Learning*, 46:131–159, 2002.

[6] R. Fergus, P. Perona, and A. Zisserman. Object class recognition by unsupervised scale-invariant learning. In *IEEE Computer Society Conference on Computer Vision and Pattern Recognition*, volume 2, 2003.

[7] R. Fergus L. Fei-Fei and P. Perona. Learning generative visual models from few training examples: an incremental bayesian approach tested on 101 object categories. In *IEEE. CVPR 2004, Workshop on Generative-Model Based Vision.*, 2004.

[8] G.R.G. Lanckriet, N. Cristianini, P. Bartlett, L. El Ghaoui, and M.I. Jordan. Learning the Kernel Matrix with Semidefinite Programming. *Journal of Machine Learning Research*, 5:27–72, 2004.

[9] Arkadi Nemirovski. Lectures on modern convex optimization (chp.5.4). Available at `www2.isye.gatech.edu/˜nemirovs/Lect_ModConvOpt.pdf`.

[10] M-E. Nilsback and A. Zisserman. A visual vocabulary for flower classification. In *Proceedings of the IEEE Conference on Computer Vision and Pattern Recognition*, 2006.

[11] M-E. Nilsback and A Zisserman. Automated flower classification over a large number of classes. In *Proceedings of the Indian Conference on Computer Vision, Graphics and Image Processing*, 2008.

[12] Maria-Elena Nilsback and Andrew Zisserman. A Visual Vocabulary for Flower Classification. In *Proceedings of the 2006 IEEE Computer Society Conference on Computer Vision and Pattern Recognition*, volume 2, pages 1447–1454, 2006.

[13] Maria-Elena Nilsback and Andrew Zisserman. Automated Flower Classification over a Large Number of Classes. In *Proceedings of the Sixth Indian Conference on Computer Vision, Graphics & Image Processing*, 2008.

[14] A. Rakotomamonjy, F. Bach, S. Canu, and Y Grandvalet. SimpleMKL. *Journal of Machine Learning Research*, 9:2491–2521, 2008.

[15] R. T. Rockafellar. *Convex Analysis*. Princeton University Press, 1970.

[16] Soren Sonnenburg, Gunnar Ratsch, Christin Schafer, and Bernhard Scholkopf. Large Scale Multiple Kernel Learning. *Journal of Machine Learning Research*, 7:1531–1565, 2006.

[17] M. Szafranski, Y. Grandvalet, and A. Rakotomamonjy. Composite Kernel Learning. In *Proceedings of the Twenty-fifth International Conference on Machine Learning (ICML)*, 2008.

[18] Vladimir Vapnik. *Statistical Learning Theory*. Wiley-Interscience, 1998.

[19] M. Varma and D. Ray. Learning the Discriminative Power Invariance Trade-off. In *Proceedings of the International Conference on Computer Vision*, 2007.

[20] Zenglin Xu, Rong Jin, Irwin King, and Michael R. Lyu. An Extended Level Method for Multiple Kernel Learning. In *Advances in Neural Information Processing Systems*, 2008.

